# Reinforcement Learning for Mixed Open-loop and Closed-loop Control

Eric A. Hansen, Andrew G. Barto, and Shlomo Zilberstein
Department of Computer Science
University of Massachusetts
Amherst, MA 01003
{hansen,barto,shlomo}@cs.umass.edu

## Abstract

Closed-loop control relies on sensory feedback that is usually assumed to be free. But if sensing incurs a cost, it may be cost-effective to take sequences of actions in open-loop mode. We describe a reinforcement learning algorithm that learns to combine open-loop and closed-loop control when sensing incurs a cost. Although we assume reliable sensors, use of open-loop control means that actions must sometimes be taken when the current state of the controlled system is uncertain. This is a special case of the hidden-state problem in reinforcement learning, and to cope, our algorithm relies on short-term memory. The main result of the paper is a rule that significantly limits exploration of possible memory states by pruning memory states for which the estimated value of information is greater than its cost. We prove that this rule allows convergence to an optimal policy.

## 1 Introduction

Reinforcement learning (RL) is widely-used for learning closed-loop control policies. Closed-loop control works well if the sensory feedback on which it relies is accurate, fast, and inexpensive. But this is not always the case. In this paper, we address problems in which sensing incurs a cost, either a direct cost for obtaining and processing sensory data or an indirect opportunity cost for dedicating limited sensors to one control task rather than another. If the cost for sensing is significant, exclusive reliance on closed-loop control may make it impossible to optimize a performance measure such as cumulative discounted reward. For such problems, we describe an RL algorithm that learns to combine open-loop and closed-loop control. By learning to take open-loop sequences of actions between sensing, it can optimize a tradeoff between the cost and value of sensing.

The problem we address is a special case of the problem of hidden state or partial observability in RL (e.g., Whitehead & Lin, 1995; McCallum, 1995). Although we assume sensing provides perfect information (a significant limiting assumption), use of open-loop control means that actions must sometimes be taken when the current state of the controlled system is uncertain. Previous work on RL for partially observable environments has focused on coping with sensors that provide imperfect or incomplete information, in contrast to deciding whether or when to sense. Tan (1991) addressed the problem of sensing costs by showing how to use RL to learn a cost-effective sensing procedure for state identification, but his work addressed the question of which sensors to use, not when to sense, and so still assumed closed-loop control.

In this paper, we formalize the problem of mixed open-loop and closed-loop control as a Markov decision process and use RL in the form of Q-learning to learn an optimal, state-dependent sensing interval. Because there is a combinatorial explosion of open-loop action sequences, we introduce a simple rule for pruning this large search space. Our most significant result is a proof that Q-learning converges to an optimal policy even when a fraction of the space of possible open-loop action sequences is explored.

## 2   Q-learning with sensing costs

Q-learning (Watkins, 1989) is a well-studied RL algorithm for learning to control a discrete-time, finite state and action Markov decision process (MDP). At each time step, a controller observes the current state $x$, takes an action $a$, and receives an immediate reward $r$ with expected value $r(x, a)$. With probability $p(x, a, y)$ the process makes a transition to state $y$, which becomes the current state on the next time step. A controller using Q-learning learns a state-action value function, $\hat{Q}(x, a)$, that estimates the expected total discounted reward for taking action $a$ in state $x$ and performing optimally thereafter. Each time step, $\hat{Q}$ is updated for state-action pair $(x, a)$ after receiving reward $r$ and observing resulting state $y$, as follows:

$$\hat{Q}(x, a) \leftarrow \hat{Q}(x, a) + \alpha \left[ r + \gamma \hat{V}(y) - \hat{Q}(x, a) \right],$$

where $\alpha \in (0, 1]$ is a learning rate parameter, $\gamma \in [0, 1)$ is a discount factor, and $\hat{V}(y) = \max_b \hat{Q}(y, b)$. Watkins and Dayan (1992) prove that $\hat{Q}$ converges to an optimal state-action value function $Q$ (and $\hat{V}$ converges to an optimal state value function $V$) with probability one if all actions continue to be tried from all states, the state-action value function is represented by a lookup-table, and the learning rate is decreased in an appropriate manner.

If there is a cost for sensing, acting optimally may require a mixed strategy of open-loop and closed-loop control that allows a controller to take open-loop sequences of actions between sensing. This possibility can be modeled by an MDP with two kinds of actions: *control actions* that have an effect on the current state but do not provide information, and a *sensing action* that reveals the current state but has no other effect. We let $o$ (for observation) denote the sensing action and assume it provides perfect information about the underlying state. Separating control actions and the sensing action gives an agent control over when to receive sensory feedback, and hence, control over sensing costs.

When one control action follows another without an intervening sensing action, the second control action is taken without knowing the underlying state. We model this by including "memory states" in the state set of the MDP. Each memory state represents memory of the last observed state and the open-loop sequence of control actions taken since; because we assume sensing provides perfect information,

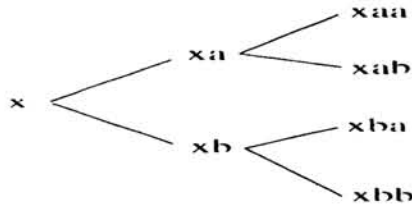

Figure 1: A tree of memory states rooted at observed state x. The set of control actions is {a,b} and the length bound is 2.

remembering this much history provides a *sufficient statistic* for action selection (Monahan, 1982). Possible memory states can be represented using a tree like the one shown in Figure 1, where the root represents the last observed state and the other nodes represent memory states, one for each possible open-loop action sequence. For example, let $xa$ denote the memory state that results from taking control action $a$ in state $x$. Similarly, let $xab$ denote the memory state that results from taking control action $b$ in memory state $xa$. Note that a control action causes a deterministic transition to a subsequent memory state, while a sensing action causes a stochastic transition to an observed state – the root of some tree. There is a tree like the one in figure 1 for each observable state.

This problem is a special case of a partially observable MDP and can be formalized in an analogous way (Monahan, 1982). Given a state-transition and reward model for a core MDP with a state set that consists only of the underlying states of a system (which for this problem we also call observable states), we can define a state-transition and reward model for an MDP that includes memory states in its state set. As a convenient notation, let $p(x, a_1..a_k, y)$ denote the probability that taking an open-loop action sequence $a_1..a_k$ from state $x$ results in state $y$, where both $x$ and $y$ are states of the underlying system. These probabilities can be computed recursively from the single-step state-transition probabilities of the core MDP as follows:

$$p(x, a_1..a_k, y) = \sum_z p(x, a_1..a_{k-1}, z)p(z, a_k, y).$$

State-transition probabilities for the sensing action can then be defined as

$$p(xa_1..a_k, o, y) = p(x, a_1..a_k, y),$$

and a reward function for the generalized MDP can be similarly defined as

$$r(xa_1..a_{k-1}, a_k) = \sum_y p(x, a_1..a_{k-1}, y)r(y, a_k),$$

where the cost of sensing in state $x$ of the core MDP is $r(x, o)$.

If we assume a bound on the number of control actions that can be taken between sensing actions (i.e., a bound on the depth of each tree) and also assume a finite number of underlying states, the number of possible memory states is finite. It follows that the MDP we have constructed is a well-defined finite state and action MDP, and all of the theory developed for Q-learning continues to apply, including its convergence proof. (This is not true of partially observable MDPs in general.) Therefore, Q-learning can in principle find an optimal policy for interleaving control actions and sensing, assuming sensing provides perfect information.

## 3    Limiting Exploration

A problem with including memory states in the state set of an MDP is that it increases the size of the state set exponentially. The combinatorial explosion of

state-action values to be learned raises doubt about the computational feasibility of this generalization of RL. We present a solution in the form of a rule for pruning each tree of memory states, thereby limiting the number of memory states that must be explored. We prove that even if some memory states are never explored, Q-learning converges to an optimal state-action value function. Because the state-action value function is left undefined for unexplored memory states, we must carefully define what we mean by an optimal state-action value function.

**Definition:** *A state-action value function is optimal if it is sufficient for generating optimal behavior and the values of the state-action pairs visited when behaving optimally are optimal.*

A state-action value function that is undefined for some states is optimal, by this definition, if a controller that follows it behaves identically to a controller with a complete, optimal state-action value function. This is possible if the states for which the state-action value function is undefined are not encountered when an agent acts optimally. Barto, Bradtke, and Singh (1995) invoke a similar idea for a different class of problems.

Let $g(xa_1..a_k)$ denote the expected reward for taking actions $a_1..a_k$ in open-loop mode after observing state $x$:

$$g(xa_1..a_k) = r(x, a_1) + \sum_{i=1}^{k-1} \gamma^i r(xa_1..a_i, a_{i+1}).$$

Let $h(xa_1..a_k)$ denote the discounted expected value of perfect information after reaching memory state $xa_1..a_k$, which is equal to the discounted Q-value for sensing in memory state $xa_1..a_k$ minus the cost for sensing in this state:

$$h(xa_1..a_k) = \gamma^k \sum_y p(xa_1..a_k, o, y) V(y) = \gamma^k (Q(xa_1..a_k, o) - r(xa_1..a_k, o)).$$

Both $g$ and $h$ are easily learned during Q-learning, and we refer to the learned estimates as $\hat{g}$ and $\hat{h}$. These are used in the pruning rule, as follows:

**Pruning rule:** *If $\hat{g}(xa_1..a_k) + \hat{h}(xa_1..a_k) \leq \hat{V}(x)$, then memory states that descend from $xa_1..a_k$ do not need to be explored. A controller should immediately execute a sensing action when it reaches one of these memory states.*

The intuition behind the pruning rule is that a branch of a tree of memory states can be pruned after reaching a memory state for which the value of information is greater than or equal to its cost. Because pruning is based on estimated values, memory states that are pruned at one point during learning may later be explored as learned estimates change. The net effect of pruning, however, is to focus exploration on a subset of memory states, and as Q-learning converges, the subset of unpruned memory states becomes stable. The following theorem is proved in an appendix.

**Theorem:** *Q-learning converges to an optimal state-action value function with probability one if, in addition to the conditions for convergence given by Watkins and Dayan (1992), exploration is limited by the pruning rule.*

This result is closely related to a similar result for solving this class of problems using dynamic programming (Hansen, 1997), where it is shown that pruning can assure convergence to an optimal policy even if no bound is placed on the length of open-loop action sequences – under the assumption that it is optimal to sense at finite intervals. This additional result can be extended to Q-learning as well, although we do not present the extension in this paper. An artificial length bound can be set as low or high as desired to ensure a finite set of memory states.

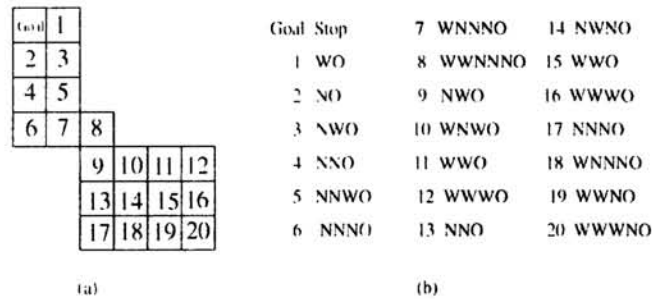

| Goal | Stop | | 7 | WNNNO | | 14 | NWNO |
|---|---|---|---|---|---|---|---|
| 1 | WO | | 8 | WWNNNO | | 15 | WWO |
| 2 | NO | | 9 | NWO | | 16 | WWWO |
| 3 | NWO | | 10 | WNWO | | 17 | NNNO |
| 4 | NNO | | 11 | WWO | | 18 | WNNNO |
| 5 | NNWO | | 12 | WWWO | | 19 | WWNO |
| 6 | NNNO | | 13 | NNO | | 20 | WWWNO |

(a)                                                  (b)

Figure 2: (a) Grid world with numbered states (b) Optimal policy

We use the notation $g$ and $h$ in our statement of the pruning rule to emphasize its relationship to pruning in heuristic search. If we regard the root of a tree of memory states as the start state and the memory state that corresponds to the best open-loop action sequence as the goal state, then $g$ can be regarded as the cost-to-arrive function and the value of perfect information $h$ can be regarded as an upper bound on the cost-to-go function.

## 4  Example

We describe a simple example to illustrate the extent of pruning possible using this rule. Imagine that a "robot" must find its way to a goal location in the upper left-hand corner of the grid shown in Figure 2a. Each cell of the grid corresponds to a state, with the states numbered for convenient reference. The robot has five control actions; it can move north, east, south, or west, one cell at a time, or it can stop. The problem ends when the robot stops. If it stops in the goal state it receives a reward of 100, otherwise it receives no reward. The robot must execute a sequence of actions to reach the goal state, but its move actions are stochastic. If the robot attempts to move in a particular direction, it succeeds with probability 0.8. With probability 0.05 it moves in a direction 90 degrees off to one side of its intended direction, with probability 0.05 it moves in a direction 90 degrees off to the other side, and with probability 0.1 it does not move at all. If the robot's movement would take it outside the grid, it remains in the same cell. Because its progress is uncertain, the robot must interleave sensing and control actions to keep track of its location. The reward for sensing is $-1$ (i.e., a cost of 1) and for each move action it is $-4$. To optimize expected total reward, the robot must find its way to the goal while minimizing the combined cost of moving and sensing.

Figure 2b shows the optimal open-loop sequence of actions for each observable state. If the bound on the length of an open-loop sequence of control actions is five, the number of possible memory states for this problem is over 64,000, a number that grows explosively as the length bound is increased (to over 16 million when the bound is nine). Using the pruning rule, Q-learning must explore just less than 1000 memory states (and no deeper than nine levels in any tree) to converge to an optimal policy, even when there is no bound on the interval between sensing actions.

## 5  Conclusion

We have described an extension of Q-learning for MDPs with sensing costs and a rule for limiting exploration that makes it possible for Q-learning to converge to an optimal policy despite exploring a fraction of possible memory states. As already pointed out, the problem we have formalized is a partially observable MDP,

although one that is restricted by the assumption that sensing provides perfect information. An interesting direction in which to pursue this work would be to explore its relationship to work on RL for partially observable MDPs, which has so far focused on the problem of sensor uncertainty and hidden state. Because some of this work also makes use of tree representations of the state space and of learned state-action values (e.g., McCallum, 1995), it may be that a similar pruning rule can constrain exploration for such problems.

## Acknowledgements

Support for this work was provided in part by the National Science Foundation under grants ECS-9214866 and IRI-9409827 and in part by Rome Laboratory, USAF, under grant F30602-95-1-0012.

## References

Barto, A.G.; Bradtke, S.J.; & Singh, S.P. (1995) Learning to act using real-time dynamic programming. *Artificial Intelligence* 72(1/2):81-138.

Hansen, E.A. (1997) Markov decision processes with observation costs. University of Massachusetts at Amherst, Computer Science Technical Report 97-01.

McCallum, R.A. (1995) Instance-based utile distinctions for reinforcement learning with hidden state. In Proc. 12th Int. Machine Learning Conf. Morgan Kaufmann.

Monahan, G.E. (1982) A survey of partially observable Markov decision processes: Theory, models, and algorithms. *Management Science* 28:1-16.

Tan, M. (1991) Cost-sensitive reinforcement learning for adaptive classification and control. In Proc. 9th Nat. Conf. on Artificial Intelligence. AAAI Press/MIT Press.

Watkins, C.J.C.H. (1989) Learning from delayed rewards. Ph.D. Thesis, University of Cambridge, England.

Watkins, C.J.C.H. & Dayan, P. (1992) Technical note: Q-learning. *Machine Learning* 8(3/4):279-292.

Whitehad, S.D. & Lin, L.-J.(1995) Reinforcement learning of non-Markov decision processes. *Artificial Intelligence* 73:271-306.

## Appendix

**Proof of theorem:** Consider an MDP with a state set that consists only of the memory states that are not pruned. We call it a "pruned MDP" to distinguish it from the original MDP for which the state set consists of all possible memory states. Because the pruned MDP is a finite state and action MDP, Q-learning with pruning converges with probability one. What we must show is that the state-action values to which it converges include every state-action pair visited by an optimal controller for the original MDP, and that for each of these state-action pairs the learned state-action value is equal to the optimal state-action value for the original MDP.

Let $\hat{Q}$ and $\hat{V}$ denote the values that are learned by Q-learning when its exploration is limited by the pruning rule, and let $Q$ and $V$ denote value functions that are optimal when the state set of the MDP includes all possible memory states. Because an MDP has an optimal stationary policy and each control action causes a deterministic transition to a subsequent memory state, there is an optimal path through each tree of memory states. The learned value of the root state of each tree is optimal if and only if the learned value of each memory state along this path is also optimal.

Therefore to show that Q-learning with pruning converges to an optimal state-action value function, it is sufficient to show that $\hat{V} = V$ for every observable state $x$. Our proof is by induction on the number of control actions that can be taken between one sensing action and the next. We use the fact that if Q-learning has converged, then $\hat{g}(xa_1..a_i) = g(xa_1..a_i)$ and $\hat{h}(xa_1..a_i) = \sum_y p(x, a_1..a_i, y)\hat{V}(y)$ for every memory state $xa_1..a_i$.

First note that if $\hat{g}(xa_1) + \gamma r(xa_1, o) + \hat{h}(xa_1) > \hat{V}(x)$, that is, if $\hat{V}$ for some observable state $x$ can be improved by exploring a path of a single control action followed by sensing, then it is contradictory to suppose Q-learning with pruning has converged because single-depth memory states in a tree are never pruned. Now, make the inductive hypothesis that Q-learning with pruning has not converged if $\hat{V}$ can be improved for some observable state by exploring a path of less than $k$ control actions before sensing. We show that it has not converged if $\hat{V}$ can be improved for some observable state by exploring a path of $k$ control actions before sensing.

Suppose $\hat{V}$ for some observable state $x$ can be improved by exploring a path that consists of taking the sequence of control actions $a_1..a_k$ before sensing, that is,

$$\hat{g}(xa_1..a_k) + \gamma^k r(xa_1..a_k, o) + \hat{h}(xa_1..a_k) > \hat{V}(x),$$

Since only pruning can prevent improvement in this case, let $xa_1..a_i$ be the memory state at which application of the pruning rule prevents $xa_1..a_k$ from being explored. Because the tree has been pruned at this node, $\hat{V}(x) \geq \hat{g}(xa_1..a_i) + \hat{h}(xa_1..a_i)$, and so

$$\hat{g}(xa_1..a_k) + \gamma^k r(xa_1..a_k, o) + \hat{h}(xa_1..a_k) > \hat{g}(xa_i..a_i) + \hat{h}(xa_1..a_i).$$

We can expand this inequality as follows:

$$\hat{g}(xa_1..a_i) + \gamma^i \sum_y p(x, a_1..a_i, y) \left[ \hat{g}(ya_{i+1}..a_k) + \gamma^{k-i} r(ya_{i+1}..a_k, o) + \hat{h}(ya_{i+1}..a_k) \right]$$

$$> \hat{g}(xa_1..a_i) + \hat{h}(xa_1..a_i).$$

Simplification and expansion of $\hat{h}$ yields

$$\sum_{y \in S} p(x, a_1..a_i, y) \left[ \hat{g}(ya_{i+1}..a_k) + \gamma^{k-i} r(ya_{i+1}..a_k, o) + \gamma^{k-i} \sum_z p(y, a_{i+1}..a_k, z)\hat{V}(z) \right]$$

$$> \sum_y p(x, a_1..a_i, y)\hat{V}(y).$$

Therefore, there is some observable state, $y$, such that

$$\hat{g}(ya_{i+1}..a_k) + \gamma^{k-i} r(ya_{i+1}..a_k, o) + \gamma^{k-i} \sum_z p(y, a_{i+1}..a_k, z)\hat{V}(z) > \hat{V}(y).$$

Because the value of observable state $y$ can be improved by taking less than $k$ control actions before sensing, by the inductive hypothesis Q-learning has not yet converged. $\square$

The proof provides insight into how pruning works. If a state-action pair along some optimal path is temporarily pruned, it must be possible to improve the value of some observable state by exploring a shorter path of memory states that has not been pruned. The resulting improvement of the value function changes the threshold for pruning and the state-action pair that was formerly pruned may no longer be so, making further improvement of the learned value function possible.